# Logarithmic Online Regret Bounds for Undiscounted Reinforcement Learning

**Peter Auer**        **Ronald Ortner**
University of Leoben, Franz-Josef-Strasse 18,
8700 Leoben, Austria
{auer,rortner}@unileoben.ac.at

## Abstract

We present a learning algorithm for undiscounted reinforcement learning. Our interest lies in bounds for the algorithm's online performance after some finite number of steps. In the spirit of similar methods already successfully applied for the exploration-exploitation tradeoff in multi-armed bandit problems, we use upper confidence bounds to show that our UCRL algorithm achieves logarithmic online regret in the number of steps taken with respect to an optimal policy.

## 1 Introduction

### 1.1 Preliminaries

**Definition 1.** *A Markov decision process (MDP) $M$ on a finite set of* states $S$ *with a finite set of* actions $A$ *available in each state* $\in S$ *consists of (i) an initial distribution $\mu_0$ over S, (ii) the transition probabilities $p(s, a, s')$ that specify the probability of reaching state $s'$ when choosing action $a$ in state $s$, and (iii) the payoff distributions with mean $r(s, a)$ and support in $[0, 1]$ that specify the random reward for choosing action $a$ in state $s$.*

A *policy* on an MDP $M$ is a mapping $\pi : S \to A$. We will mainly consider *unichain* MDPs, in which under any policy any state can be reached (after a finite number of transitions) from any state. For a policy $\pi$ let $\mu_\pi$ be the stationary distribution induced by $\pi$ on $M$.[1] The *average reward* of $\pi$ then is defined as

$$\rho(M, \pi) := \sum_{s \in S} \mu_\pi(s) r(s, \pi(s)). \tag{1}$$

A policy $\pi^*$ is called *optimal* on $M$, if $\rho(M, \pi) \leq \rho(M, \pi^*) =: \rho^*(M) =: \rho^*$ for all policies $\pi$.

Our measure for the quality of a learning algorithm is the total regret after some finite number of steps. When a learning algorithm $\mathcal{A}$ executes action $a_t$ in state $s_t$ at step $t$ obtaining reward $r_t$, then $R_T := \sum_{t=0}^{T-1} r_t - T\rho^*$ denotes the *total regret of $\mathcal{A}$ after $T$ steps. The total regret $R_T^\varepsilon$ with respect to an $\varepsilon$-optimal policy* (i.e. a policy whose return differs from $\rho^*$ by at most $\varepsilon$) is defined accordingly.

### 1.2 Discussion

We would like to compare this approach with the various PAC-like bounds in the literature as given for the E³-algorithm of Kearns, Singh [1] and the R-Max algorithm of Brafman, Tennenholtz [2] (cf. also [3]). Both take as inputs (among others) a confidence parameter $\delta$ and an accuracy parameter

$\varepsilon$. The algorithms then are shown to yield $\varepsilon$-optimal return after time polynomial in $\frac{1}{\delta}$, $\frac{1}{\varepsilon}$ (among others) with probability $1 - \delta$. In contrast, our algorithm has no such input parameters and converges to an optimal policy with expected logarithmic online regret in the number of steps taken.

Obviously, by using a decreasing sequence $\varepsilon_t$, online regret bounds for E³ and R-Max can be achieved. However, it is not clear whether such a procedure can give logarithmic online regret bounds. We rather conjecture that these bounds either will be not logarithmic in the total number of steps (if $\varepsilon_t$ decreases quickly) or that the dependency on the parameters of the MDP – in particular on the distance between the reward of the best and a second best policy – won't be polynomial (if $\varepsilon_t$ decreases slowly).

Moreover, although our UCRL algorithm shares the "optimism under uncertainty" maxim with R-Max, our mechanism for the exploitation-exploration tradeoff is implicit, while E³ and R-Max have to distinguish between "known" and "unknown" states explicitly. Finally, in their original form both E³ and R-Max need a policy's $\varepsilon$-return mixing time $T_\varepsilon$ as input parameter. The knowledge of this parameter then is eliminated by calculating the $\varepsilon$-optimal policy for $T_\varepsilon = 1, 2, \ldots$, so that sooner or later the correct $\varepsilon$-return mixing time is reached. This is sufficient to obtain polynomial PAC-bounds, but seems to be intricate for practical purposes. Moreover, as noted in [2], at some time step the assumed $T_\varepsilon$ may be exponential in the true $T_\varepsilon$, which makes policy computation exponential in $T_\varepsilon$. Unlike that, we need our mixing time parameter only in the analysis. This makes our algorithm rather simple and intuitive.

Recently, more refined performance measures such as the *sample complexity of exploration* [3] were introduced. Strehl and Littman [4] showed that in the discounted setting, efficiency in the sample complexity implies efficiency in the *average loss*. However, *average loss* is defined in respect to the actually visited states, so that small *average* loss does not guarantee small *total* regret, which is defined in respect to the states visited by an optimal policy. For this average loss polylogarithmic online bounds were shown for for the MBIE algorithm [4], while more recently logarithmic bounds for delayed Q-learning were given in [5]. However, discounted reinforcement learning is a bit simpler than undiscounted reinforcement learning, as depending on the discount factor only a finite number of steps is relevant. This makes discounted reinforcement learning similar to the setting with trials of constant length from a fixed initial state [6]. For this case logarithmic online regret bounds in the number of trials have already been given in [7].

Since we measure performance during exploration, the exploration vs. exploitation dilemma becomes an important issue. In the multi-armed bandit problem, similar exploration-exploitation tradeoffs were handled with upper confidence bounds for the expected immediate returns [8, 9]. This approach has been shown to allow good performance during the learning phase, while still converging fast to a nearly optimal policy. Our UCRL algorithm takes into account the state structure of the MDP, but is still based on upper confidence bounds for the expected return of a policy. Upper confidence bounds have been applied to reinforcement learning in various places and different contexts, e.g. interval estimation [10, 11], action elimination [12], or PAC-learning [6]. Our UCRL algorithm is similar to Strehl, Littman's MBIE algorithm [10, 4], but our confidence bounds are different, and we are interested in the undiscounted case.

Another paper with a similar approach is Burnetas, Katehakis [13]. The basic idea of their rather complex *index policies* is to choose the action with maximal return in some specified confidence region of the MDP's probability distributions. The online-regret of their algorithm is asymptotically logarithmic in the number of steps, which is best possible. Our UCRL algorithm is simpler and achieves logarithmic regret not only asymptotically but uniformly over time. Moreover, unlike in the approach of [13], knowledge about the MDP's underlying state structure is not needed.

More recently, online reinforcement learning with changing rewards chosen by an adversary was considered under the presumption that the learner has full knowledge of the transition probabilities [14]. The given algorithm achieves best possible regret of $O(\sqrt{T})$ after $T$ steps.

In the subsequent Sections 2 and 3 we introduce our UCRL algorithm and show that its expected online regret in unichain MDPs is $O(\log T)$ after $T$ steps. In Section 4 we consider problems that arise when the underlying MDP is not unichain.

## 2    The UCRL Algorithm

To select good policies, we keep track of estimates for the average rewards and the transition probabilities. For each step $t$ let

$$
\begin{aligned}
N_t(s,a) &= |\{0 \leq \tau < t : s_\tau = s, a_\tau = a\}|, \\
R_t(s,a) &= \sum_{\substack{0 \leq \tau < t: \\ s_\tau = s, a_\tau = a}} r_\tau, \\
P_t(s,a,s') &= |\{0 \leq \tau < t : s_\tau = s, a_\tau = a, s_{\tau+1} = s'\}|,
\end{aligned}
$$

be the number of steps when action $a$ was chosen in state $s$, the sum of rewards obtained when choosing this action, and the number of times the transition was to state $s'$, respectively. From these numbers we immediately get estimates for the average rewards and transition probabilities,

$$
\hat{r}_t(s,a) := \frac{R_t(s,a)}{N_t(s,a)},
$$

$$
\hat{p}_t(s,a,s') := \frac{P_t(s,a,s')}{N_t(s,a)},
$$

provided that the *number of visits in* $(s,a)$, $N_t(s,a) > 0$. In general, these estimates will deviate from the respective true values. However, together with appropriate confidence intervals they may be used to define a set $\mathcal{M}_t$ of plausible MDPs. Our algorithm then chooses an optimal policy $\tilde{\pi}_t$ for an MDP $\tilde{M}_t$ with maximal average reward $\tilde{\rho}_t^* := \rho^*(\tilde{M}_t)$ among the MDPs in $\mathcal{M}_t$. That is,

$$
\begin{aligned}
\tilde{\pi}_t &:= \arg\max_\pi \{\rho(M,\pi) : M \in \mathcal{M}_t\}, \quad \text{and} \\
\tilde{M}_t &:= \arg\max_{M \in \mathcal{M}_t} \{\rho(M, \tilde{\pi}_t)\}.
\end{aligned}
$$

More precisely, we want $\mathcal{M}_t$ to be a set of plausible MDPs in the sense that

$$
\mathbb{P}\{\rho^* > \tilde{\rho}_t^*\} < t^{-\alpha} \tag{2}
$$

for some $\alpha > 2$. Essentially, condition (2) means that it is unlikely that the true MDP $M$ is not in $\mathcal{M}_t$. Actually, $\mathcal{M}_t$ is defined to contain exactly those *unichain* MDPs $M'$ whose transition probabilities $p'(\cdot,\cdot,\cdot)$ and rewards $r'(\cdot,\cdot)$ satisfy for all states $s$, $s'$ and actions $a$

$$
r'(s,a) \leq \hat{r}_t(s,a) + \sqrt{\tfrac{\log(2t^\alpha |S||A|)}{2N_t(s,a)}}, \quad \text{and} \tag{3}
$$

$$
|p'(s,a,s') - \hat{p}_t(s,a,s')| \leq \sqrt{\tfrac{\log(4t^\alpha |S|^2|A|)}{2N_t(s,a)}}. \tag{4}
$$

Conditions (3) and (4) describe confidence bounds on the rewards and transition probabilities of the true MDP $M$ such that (2) is implied (cf. Section 3.1 below). The intuition behind the algorithm is that if a non-optimal policy is followed, then this is eventually observed and something about the MDP is learned. In the proofs we show that this learning happens sufficiently fast to approach an optimal policy with only logarithmic regret.

As switching policies too often may be harmful, and estimates don't change very much after few steps, our algorithm discards the policy $\tilde{\pi}_t$ only if there was considerable progress concerning the estimates $\hat{p}(s, \tilde{\pi}_t(s), s')$ or $\hat{r}(s, \tilde{\pi}_t(s))$. That is, UCRL sticks to a policy until the length of some of the confidence intervals given by conditions (3) and (4) is halved. Only then a new policy is calculated. We will see below (cf. Section 3.3) that this condition limits the number of policy changes without paying too much for not changing to an optimal policy earlier. Summing up, Figure 1 displays our algorithm.

**Remark 1.** *The optimal policy $\tilde{\pi}$ in the algorithm can be efficiently calculated by a modified version of value iteration (cf. [15]).*

## 3    Analysis for Unichain MDPs

### 3.1    An Upper Bound on the Optimal Reward

We show that with high probability the true MDP $M$ is contained in the set $\mathcal{M}_t$ of plausible MDPs.

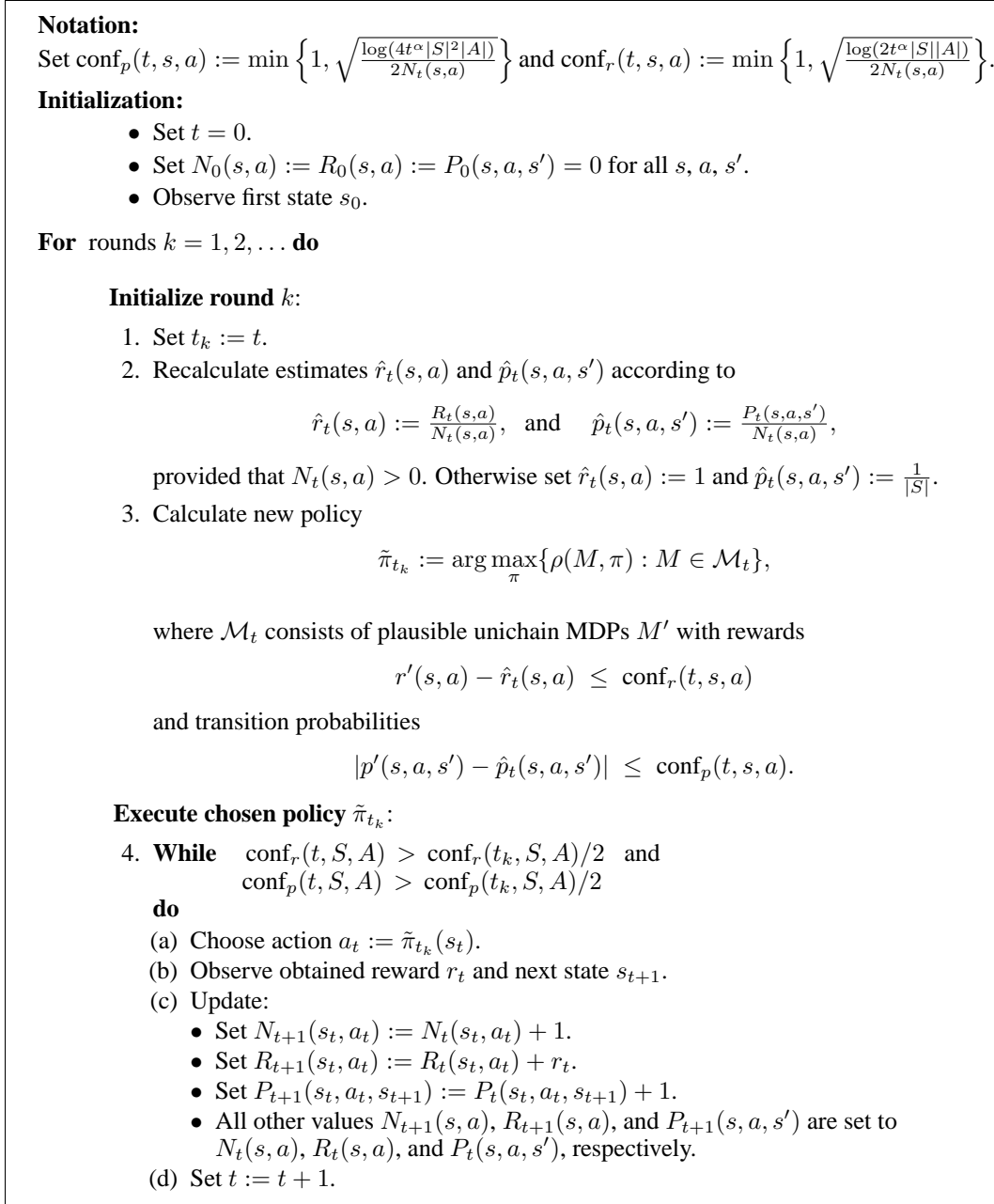

**Notation:**
Set $\mathrm{conf}_p(t,s,a) := \min\left\{1, \sqrt{\frac{\log(4t^\alpha|S|^2|A|)}{2N_t(s,a)}}\right\}$ and $\mathrm{conf}_r(t,s,a) := \min\left\{1, \sqrt{\frac{\log(2t^\alpha|S||A|)}{2N_t(s,a)}}\right\}$.

**Initialization:**
- Set $t = 0$.
- Set $N_0(s,a) := R_0(s,a) := P_0(s,a,s') = 0$ for all $s$, $a$, $s'$.
- Observe first state $s_0$.

**For** rounds $k = 1, 2, \ldots$ **do**

    **Initialize round** $k$:

1. Set $t_k := t$.
2. Recalculate estimates $\hat{r}_t(s,a)$ and $\hat{p}_t(s,a,s')$ according to
$$\hat{r}_t(s,a) := \frac{R_t(s,a)}{N_t(s,a)}, \quad \text{and} \quad \hat{p}_t(s,a,s') := \frac{P_t(s,a,s')}{N_t(s,a)},$$
   provided that $N_t(s,a) > 0$. Otherwise set $\hat{r}_t(s,a) := 1$ and $\hat{p}_t(s,a,s') := \frac{1}{|S|}$.
3. Calculate new policy
$$\tilde{\pi}_{t_k} := \arg\max_\pi\{\rho(M,\pi) : M \in \mathcal{M}_t\},$$
   where $\mathcal{M}_t$ consists of plausible unichain MDPs $M'$ with rewards
$$r'(s,a) - \hat{r}_t(s,a) \;\leq\; \mathrm{conf}_r(t,s,a)$$
   and transition probabilities
$$|p'(s,a,s') - \hat{p}_t(s,a,s')| \;\leq\; \mathrm{conf}_p(t,s,a).$$

    **Execute chosen policy** $\tilde{\pi}_{t_k}$:

4. **While** $\quad \mathrm{conf}_r(t,S,A) > \mathrm{conf}_r(t_k,S,A)/2 \quad$ and
   $\qquad\qquad \mathrm{conf}_p(t,S,A) > \mathrm{conf}_p(t_k,S,A)/2$
   **do**
   (a) Choose action $a_t := \tilde{\pi}_{t_k}(s_t)$.
   (b) Observe obtained reward $r_t$ and next state $s_{t+1}$.
   (c) Update:
      - Set $N_{t+1}(s_t,a_t) := N_t(s_t,a_t) + 1$.
      - Set $R_{t+1}(s_t,a_t) := R_t(s_t,a_t) + r_t$.
      - Set $P_{t+1}(s_t,a_t,s_{t+1}) := P_t(s_t,a_t,s_{t+1}) + 1$.
      - All other values $N_{t+1}(s,a)$, $R_{t+1}(s,a)$, and $P_{t+1}(s,a,s')$ are set to $N_t(s,a)$, $R_t(s,a)$, and $P_t(s,a,s')$, respectively.
   (d) Set $t := t + 1$.

Figure 1: The UCRL algorithm.

**Lemma 1.** *For any $t$, any reward $r(s,a)$ and any transition probability $p(s,a,s')$ of the true MDP $M$ we have*

$$\mathbb{P}\left\{\hat{r}_t(s,a) < r(s,a) - \sqrt{\frac{\log(2t^\alpha|S||A|)}{2N_t(s,a)}}\right\} \;<\; \frac{t^{-\alpha}}{2|S||A|}, \tag{5}$$

$$\mathbb{P}\left\{|\hat{p}_t(s,a,s') - p(s,a,s')| > \sqrt{\frac{\log(4t^\alpha|S|^2|A|)}{2N_t(s,a)}}\right\} \;<\; \frac{t^{-\alpha}}{2|S|^2|A|}. \tag{6}$$

*Proof.* By Chernoff-Hoeffding's inequality. $\qquad\qquad\qquad\qquad\qquad\qquad\qquad\qquad\qquad$ $\square$

Using the definition of $\mathcal{M}_t$ as given by (3) and (4) and summing over all $s$, $a$, and $s'$, Lemma 1 shows that $M \in \mathcal{M}_t$ with high probability. This implies that the maximal average reward $\tilde{\rho}_t^*$ assumed by our algorithm when calculating a new policy at step $t$ is an upper bound on $\rho^*(M)$ with high probability.

**Corollary 1.** *For any $t$:* $\quad \mathbb{P}\{\rho^* > \tilde{\rho}_t^*\} < t^{-\alpha}$.

### 3.2 Sufficient Precision and Mixing Times

In order to upper bound the loss, we consider the precision needed to guarantee that the policy calculated by UCRL is ($\varepsilon$-)optimal. This sufficient precision will of course depend on $\varepsilon$ or – in case one wants to compete with an optimal policy – the minimal difference between $\rho^*$ and the average reward of some suboptimal policy,

$$\Delta := \min_{\pi:\rho(M,\pi)<\rho^*} \rho^* - \rho(M,\pi).$$

It is sufficient that the difference between $\rho(\tilde{M}_t, \tilde{\pi}_t)$ and $\rho(M, \tilde{\pi}_t)$ is small in order to guarantee that $\tilde{\pi}_t$ is an ($\varepsilon$-)optimal policy. For if $|\rho(\tilde{M}_t, \tilde{\pi}_t) - \rho(M, \tilde{\pi}_t)| < \varepsilon$, then by Corollary 1 with high probability

$$\varepsilon > |\rho(\tilde{M}_t, \tilde{\pi}_t) - \rho(M, \tilde{\pi}_t)| \geq |\rho^*(M) - \rho(M, \tilde{\pi}_t)|, \tag{7}$$

so that $\tilde{\pi}_t$ is already an $\varepsilon$-optimal policy on $M$. For $\varepsilon = \Delta$, (7) implies the optimality of $\tilde{\pi}_t$.

Thus, we consider bounds on the deviation of the transition probabilities and rewards for the assumed MDP $\tilde{M}_t$ from the true values, such that (7) is implied. This is handled in the subsequent proposition, where we use the notion of the MDP's *mixing time*, which will play an essential role throughout the analysis.

**Definition 2.** *Given an ergodic Markov chain $C$, let $T_{s,s'}$ be the* first passage time *for two states $s$, $s'$, that is, the time needed to reach $s'$ when starting in $s$. Furthermore let $T_{s,s}$ the* return time *to state $s$. Let $T_C := \max_{s,s' \in S} \mathbb{E}(T_{s,s'})$, and $\kappa_C := \max_{s \in S} \frac{\max_{s' \neq s} \mathbb{E}(T_{s',s})}{2\mathbb{E}(T_{s,s})}$. Then the* mixing time *of a unichain MDP $M$ is $T_M := \max_\pi T_{C_\pi}$, where $C_\pi$ is the Markov chain induced by $\pi$ on $M$. Furthermore, we set $\kappa_M := \max_\pi \kappa_{C_\pi}$.*

Our notion of *mixing time* is different from the notion of *$\varepsilon$-return mixing time* given in [1, 2], which depends on an additional parameter $\varepsilon$. However, it serves a similar purpose.

**Proposition 1.** *Let $p(\cdot,\cdot)$, $\tilde{p}(\cdot,\cdot)$ and $r(\cdot)$, $\tilde{r}(\cdot)$ be the transition probabilities and rewards of the MDPs $M$ and $\tilde{M}$ under the policy $\tilde{\pi}$, respectively. If for all states $s, s'$*

$$|\tilde{r}(s) - r(s)| < \varepsilon_r := \frac{\varepsilon}{2} \quad and \quad |\tilde{p}(s,s') - p(s,s')| < \varepsilon_p := \frac{\varepsilon}{2\kappa_M|S|^2},$$

*then $|\rho(\tilde{M}, \tilde{\pi}) - \rho(M, \tilde{\pi})| < \varepsilon$.*

The proposition is an easy consequence of the following result about the difference in the stationary distributions of ergodic Markov chains.

**Theorem 1 (Cho, Meyer[16]).** *Let $C$, $\tilde{C}$ be two ergodic Markov chains on the same state space $S$ with transition probabilities $p(\cdot,\cdot)$, $\tilde{p}(\cdot,\cdot)$ and stationary distributions $\mu$, $\tilde{\mu}$. Then the difference in the distributions $\mu$, $\tilde{\mu}$ can be upper bounded by the difference in the transition probabilities as follows:*

$$\max_{s \in S} |\mu(s) - \tilde{\mu}(s)| \leq \kappa_C \max_{s \in S} \sum_{s' \in S} |p(s,s') - \tilde{p}(s,s')|, \tag{8}$$

*where $\kappa_C$ is as given in Definition 2.*

*Proof of Proposition 1.* By (8),

$$\sum_{s \in S} |\mu(s) - \tilde{\mu}(s)| \leq |S|\kappa_M \max_{s \in S} \sum_{s' \in S} |\tilde{p}(s,s') - p(s,s')| \leq \kappa_M|S|^2\varepsilon_p.$$

As the rewards are $\in [0, 1]$ and $\sum_s \mu(s) = 1$, we have by (1)

$$
\begin{aligned}
|\rho(\tilde{M}, \tilde{\pi}) - \rho(M, \tilde{\pi})| &\leq \sum_{s \in S} |\tilde{\mu}(s) - \mu(s)| \tilde{r}(s) + \sum_{s \in S} |\tilde{r}(s) - r(s)| \mu(s) \\
&< \kappa_M |S|^2 \varepsilon_p + \varepsilon_r = \varepsilon. \qquad \Box
\end{aligned}
$$

Since $\varepsilon_r > \varepsilon_p$ and the confidence intervals for rewards are smaller than for transition probabilities (cf. Lemma 1), in the following we only consider the precision needed for transition probabilities.

## 3.3 Bounding the Regret

As can be seen from the description of the algorithm, we split the sequence of steps into *rounds*, where a new round starts whenever the algorithm recalculates its policy. The following facts follow immediately from the form of our confidence intervals and Lemma 1, respectively.

**Proposition 2.** *For halving a confidence interval of a reward or transition probability for some $(s, a) \in S \times A$, the number $N_t(s, a)$ of visits in $(s, a)$ has to be at least doubled.*

**Corollary 2.** *The number of rounds after $T$ steps cannot exceed $|S||A| \log_2 \frac{T}{|S||A|}$.*

**Proposition 3.** *If $N_t(s, a) \geq \frac{\log(4t^\alpha |S|^2 |A|)}{2\theta^2}$, then the confidence intervals for $(s, a)$ are smaller than $\theta$.*

We need to consider three sources of regret: first, by executing a suboptimal policy in a round of length $\tau$, we may lose reward up to $\tau$ within this round; second, there may be some loss when changing policies; third, we have to consider the error probabilities with which some of our confidence intervals fail.

### 3.3.1 Regret due to Suboptimal Rounds

Proposition 3 provides an upper bound on the number of visits needed in each $(s, a)$ in order to guarantee that a newly calculated policy is optimal. This can be used to upper bound the total number of steps in suboptimal rounds.

Consider all suboptimal rounds with $|\hat{p}_{t_k}(s, a, s') - p(s, a, s')| \geq \varepsilon_p$ for some $s'$, where a policy $\tilde{\pi}_{t_k}$ with $\tilde{\pi}_{t_k}(s) = a$ is played. Let $m(s, a)$ be the number of these rounds and $\tau_i(s, a)$ ($i = 1, \ldots, m(s, a)$) their respective lengths. The mean passage time between any state $s''$ and $s$ is upper bounded by $T_M$. Then by Markov's inequality, the probability that it takes more than $2T_M$ steps to reach $s$ from $s''$ is smaller than $\frac{1}{2}$. Thus we may separate each round $i$ into $\lfloor \frac{\tau_i(s,a)}{2T_M} \rfloor$ intervals of length $\geq 2T_M$, in each of which the probability of visiting state $s$ is at least $\frac{1}{2}$. Thus we may lower bound the number of visits $N_{s,a}(n)$ in $(s, a)$ within $n$ such intervals by an application of Chernoff-Hoeffding's inequality:

$$
\mathbb{P}\left\{ N_{s,a}(n) \geq \frac{n}{2} - \sqrt{n \log T} \right\} \geq 1 - \frac{1}{T}. \tag{9}
$$

Since by Proposition 3, $N_t(s, a) < 2\frac{\log(4T^\alpha |S|^2 |A|)}{\varepsilon_p 2}$, we get

$$
\sum_{i=1}^{m(s,a)} \left\lfloor \frac{\tau_i(s, a)}{2T_M} \right\rfloor < c \frac{\log(4T^\alpha |S|^2 |A|)}{\varepsilon_p 2}
$$

with probability $1 - \frac{1}{T}$ for a suitable constant $c < 11$. This gives for the expected regret in these rounds

$$
\mathbb{E}\left( \sum_{i=1}^{m(s,a)} \tau_i(s, a) \right) < 2c \cdot T_M \frac{\log(4T^\alpha |S|^2 |A|)}{\varepsilon_p 2} + 2m(s, a)T_M + \frac{1}{T}T.
$$

Applying Corollary 2 and summing up over all $(s, a)$, one sees that the expected regret due to suboptimal rounds cannot exceed

$$
2c|S||A|T_M \frac{\log(4T^\alpha |S|^2 |A|)}{\varepsilon_p 2} + 2T_M |S|^2 |A|^2 \log_2 \frac{T}{|S||A|} + |S||A|.
$$

**3.3.2 Loss by Policy Changes** For any policy $\tilde{\pi}_t$ there may be some states from which the expected average reward for the next $\tau$ steps is larger than when starting in some other state. This does not play a role if $\tau \to \infty$. However, as we are playing our policies only for a finite number of steps before considering a change, we have to take into account that every time we switch policies, we may need a start-up phase to get into such a favorable state. In average, this cannot take more than $T_M$ steps, as this time is sufficient to reach any "good" state from some "bad" state. This is made more precise in the following lemma. We omit a detailed proof.

**Lemma 2.** *For all policies $\pi$, all starting states $s_0$ and all $T \geq 0$*

$$\mathbb{E}\Big( \sum_{t=0}^{T-1} r(s_t, \pi(s_t)) \Big) \geq T\rho(\pi, M) - T_M.$$

By Corollary 2, the corresponding expected regret after $T$ steps is $\leq |S||A|T_M \log_2 \frac{T}{|S||A|}$.

**3.3.3 Regret if Confidence Intervals Fail** Finally, we have to take into account the error probabilities, with which in each round a transition probability or a reward, respectively, is not contained in its confidence interval. According to Lemma 1, the probability that this happens at some step $t$ for a given state-action pair is $< \frac{t^{-\alpha}}{2|S||A|} + |S|\frac{t^{-\alpha}}{2|S|^2|A|} = \frac{t^{-\alpha}}{|S||A|}$. Now let $t_1 = 1, t_2, \ldots, t_N \leq T$ be the steps in which a new round starts. As the regret in each round can be upper bounded by its length, one obtains for the regret caused by failure of confidence intervals

$$\sum_{i=1}^{N-1} \frac{t_i^{-\alpha}}{|S||A|}(t_{i+1} - t_i) \leq \sum_{i=1}^{N-1} \frac{t_i^{-\alpha}}{|S||A|} ct_i < \sum_{t=1}^{\infty} c\frac{t^{1-\alpha}}{|S||A|} < c',$$

using that $t_{i+1} - t_i < ct_i$ for a suitable constant $c = c(|S|, |A|, T_M)$ and provided that $\alpha > 2$.

**3.3.4 Putting Everything Together** Summing up over all the sources of regret and replacing for $\varepsilon_p$ yields the following theorem, which is a generalization of similar results that were achieved for the multi-armed bandit problem in [8].

**Theorem 2.** *On unichain MDPs, the expected total regret of the UCRL algorithm with respect to an $(\varepsilon\text{-})$optimal policy after $T > 1$ steps can be upper bounded by*

$$\mathbb{E}(R_T^\varepsilon) < const \cdot \frac{|A|T_M\kappa_M^2|S|^5}{\varepsilon^2} \log T + 3T_M|S|^2|A|^2 \log_2 \frac{T}{|S||A|}, \quad and$$

$$\mathbb{E}(R_T) < const \cdot \frac{|A|T_M\kappa_M^2|S|^5}{\Delta^2} \log T + 3T_M|S|^2|A|^2 \log_2 \frac{T}{|S||A|}.$$

# 4 Remarks and Open Questions on Multichain MDPs

In a *multichain* MDP a policy $\pi$ may split up the MDP into ergodic subchains $S_i^\pi$. Thus it may happen during the learning phase that one goes wrong and ends up in a part of the MDP that gives suboptimal return but cannot be left under no policy whatsoever. As already observed by Kearns, Singh [1], in this case it seems fair to compete with $\rho^*(M) := \max_\pi \min_{S_i^\pi} \rho(S_i^\pi, \pi)$.

Unfortunately, the original UCRL algorithm may not work very well in this setting, as it is impossible for the algorithm to distinguish between a very low probability for a transition and its impossibility. Here the "optimism in the face of uncertainty" idea fails, as there is no way to falsify the wrong belief in a possible transition.

Obviously, if we knew for each policy which subchains it induces on $M$ (the MDP's *ergodic structure*), UCRL could choose an MDP $\tilde{M}_t$ and a policy $\tilde{\pi}_t$ that maximizes the reward among all plausible MDPs with the given ergodic structure. However, only the *empiric ergodic structure* (based on the observations so far) is known. As the empiric ergodic structure may not be reliable, one may additionally explore the ergodic structures of all policies. Alas, the number of additional exploration steps will depend on the smallest positive transition probability. If the latter is not known, it seems that logarithmic online regret bounds can be no longer guaranteed.

However, we conjecture that for a slightly modified algorithm the logarithmic online regret bounds still hold for *communicating* MDPs, in which for any two states $s, s'$ there is a suitable policy $\pi$ such that $s$ is reachable from $s'$ under $\pi$ (i.e., $s, s'$ are contained in the same subchain $S_i^\pi$). As Theorem 1 does not hold for communicating MDPs in general, a proof would need a different analysis.

## 5 Conclusion and Outlook

Beside the open problems on multichain MDPs, it is an interesting question whether our results also hold when assuming for the mixing time not the slowest policy for reaching any state but the fastest. Another research direction is to consider value function approximation and continuous reinforcement learning problems.

For practical purposes, using the variance of the estimates will reduce the width of the upper confidence bounds and will make the exploration even more focused, improving learning speed and regret bounds. In this setting, we have experimental results comparable to those of the MBIE algorithm [10], which clearly outperforms other learning algorithms like R-Max or $\varepsilon$-greedy.

**Acknowledgements.**

This work was supported in part by the the Austrian Science Fund FWF (S9104-N04 SP4) and the IST Programme of the European Community, under the PASCAL Network of Excellence, IST-2002-506778. This publication only reflects the authors' views.

## Footnotes

[1]Every policy $\pi$ induces a Markov chain $C_\pi$ on $M$. If $C_\pi$ is ergodic with transition matrix $P$, then there exists a unique invariant and strictly positive distribution $\mu_\pi$, such that independent of $\mu_0$ one has $\mu_n = \mu_0 \bar{P}_n \to \mu_\pi$, where $\bar{P}_n = \frac{1}{n} \sum_{j=1}^n P^j$. If $C_\pi$ is not ergodic, $\mu_\pi$ will depend on $\mu_0$.

## References

[1] Michael J. Kearns and Satinder P. Singh. Near-optimal reinforcement learning in polynomial time. *Mach. Learn.*, 49:209–232, 2002.

[2] Ronen I. Brafman and Moshe Tennenholtz. R-max – a general polynomial time algorithm for near-optimal reinforcement learning. *J. Mach. Learn. Res.*, 3:213–231, 2002.

[3] Sham M. Kakade. *On the Sample Complexity of Reinforcement Learning*. PhD thesis, University College London, 2003.

[4] Alexander L. Strehl and Michael L. Littman. A theoretical analysis of model-based interval estimation. In *Proc. 22nd ICML 2005*, pages 857–864, 2005.

[5] Alexander L. Strehl, Lihong Li, Eric Wiewiora, John Langford, and Michael L. Littman. Pac model-free reinforcement learning. In *Proc. 23nd ICML 2006*, pages 881–888, 2006.

[6] Claude-Nicolas Fiechter. Efficient reinforcement learning. In *Proc. 7th COLT*, pages 88–97. ACM, 1994.

[7] Peter Auer and Ronald Ortner. Online regret bounds for a new reinforcement learning algorithm. In *Proc. 1st ACVW*, pages 35–42. ÖCG, 2005.

[8] Peter Auer. Using confidence bounds for exploitation-exploration trade-offs. *J. Mach. Learn. Res.*, 3:397–422, 2002.

[9] Peter Auer, Nicolò Cesa-Bianchi, and Paul Fischer. Finite-time analysis of the multi-armed bandit problem. *Mach. Learn.*, 47:235–256, 2002.

[10] Alexander L. Strehl and Michael L. Littman. An empirical evaluation of interval estimation for Markov decision processes. In *Proc. 16th ICTAI*, pages 128–135. IEEE Computer Society, 2004.

[11] Leslie P. Kaelbling. *Learning in Embedded Systems*. MIT Press, 1993.

[12] Eyal Even-Dar, Shie Mannor, and Yishay Mansour. Action elimination and stopping conditions for reinforcement learning. In *Proc. 20th ICML*, pages 162–169. AAAI Press, 2003.

[13] Apostolos N. Burnetas and Michael N. Katehakis. Optimal adaptive policies for Markov decision processes. *Math. Oper. Res.*, 22(1):222–255, 1997.

[14] Eyal Even-Dar, Sham M. Kakade, and Yishay Mansour. Experts in a Markov decision process. In *Proc. 17th NIPS*, pages 401–408. MIT Press, 2004.

[15] Martin L. Puterman. *Markov Decision Processes. Discrete Stochastic Programming*. Wiley, 1994.

[16] Grace E. Cho and Carl D. Meyer. Markov chain sensitivity measured by mean first passage times. *Linear Algebra Appl.*, 316:21–28, 2000.
